# Nonparametric Transforms of Graph Kernels for Semi-Supervised Learning

**Xiaojin Zhu**[†]    **Jaz Kandola**[‡]    **Zoubin Ghahramani**[‡†]    **John Lafferty**[†]

[†]School of Computer Science
Carnegie Mellon University
5000 Forbes Avenue
Pittsburgh, PA 15213 USA

[‡]Gatsby Computational Neuroscience Unit
University College London
17 Queen Square
London WC1N 3AR UK

## Abstract

We present an algorithm based on convex optimization for constructing kernels for semi-supervised learning. The kernel matrices are derived from the spectral decomposition of graph Laplacians, and combine labeled and unlabeled data in a systematic fashion. Unlike previous work using diffusion kernels and Gaussian random field kernels, a nonparametric kernel approach is presented that incorporates order constraints during optimization. This results in flexible kernels and avoids the need to choose among different parametric forms. Our approach relies on a quadratically constrained quadratic program (QCQP), and is computationally feasible for large datasets. We evaluate the kernels on real datasets using support vector machines, with encouraging results.

## 1  Introduction

Semi-supervised learning has been the focus of considerable recent research. In this learning problem the data consist of a set of points, with some of the points labeled and the remaining points unlabeled. The task is to use the unlabeled data to improve classification performance. Semi-supervised methods have the potential to improve many real-world problems, since unlabeled data are often far easier to obtain than labeled data.

Kernel-based methods are increasingly being used for data modeling and prediction because of their conceptual simplicity and good performance on many tasks. A promising family of semi-supervised learning methods can be viewed as constructing kernels by transforming the spectrum of a "local similarity" graph over labeled and unlabeled data. These kernels, or regularizers, penalize functions that are not smooth over the graph [7]. Informally, a smooth eigenvector has the property that two elements of the vector have similar values if there are many large weight paths between them on the graph. This results in the desirable behavior of the labels varying smoothly over the graph, as sought by, e.g., spectral clustering approaches [2], diffusion kernels [5], and the Gaussian random field approach [9]. However, the modification to the spectrum, called a spectral transformation, is often a function chosen from some parameterized family. As examples, for the diffusion kernel the spectral transformation is an exponential function, and for the Gaussian field kernel the transformation is a smoothed inverse function.

In using a parametric approach one faces the difficult problem of choosing an appropriate family of spectral transformations. For many familes the number of degrees of freedom in the parameterization may be insufficient to accurately model the data. In this paper

we propose an effective nonparametric method to find an optimal spectral transformation using kernel alignment. The main advantage of using kernel alignment is that it gives us a convex optimization problem, and does not suffer from poor convergence to local minima. A key assumption of a spectral transformation is monotonicity, so that unsmooth functions over the data graph are penalized more severly. We realize this property by imposing order constraints. The optimization problem in general is solved using semi-definite programming (SDP) [1]; however, in our approach the problem can be formulated in terms of quadratically constrained quadratic programming (QCQP), which can be solved more efficiently than a general SDP.

This paper is structured as follows. In Section 2 we review some graph theoretic concepts and relate them to the construction of kernels for semi-supervised learning. In Section 3 we introduce convex optimization via QCQP and relate it to the more familiar linear and quadratic programming commonly used in machine learning. Section 4 poses the problem of kernel based semi-supervised learning as a QCQP problem with order constraints. Experimental results using the proposed optimization framework are presented in Section 5. The results indicate that the semi-supervised kernels constructed from the learned spectral transformations perform well in practice.

## 2   Semi-supervised Kernels from Graph Spectra

We are given a labeled dataset consisting of input-output pairs $\{(x_1, y_1), \ldots, (x_l, y_l)\}$ and a (typically much larger) unlabeled dataset $\{x_{l+1}, \ldots, x_n\}$ where $x$ is in some general input space and $y$ is potentially from multiple classes. Our objective is to construct a kernel that is appropriate for the classification task. Since our methods use both the labeled and unlabeled data, we will refer to the resulting kernels as *semi-supervised kernels*. More specifically, we restrict ourselves to the *transductive* setting where the unlabeled data also serve as the test data. As such, we only need to find a good Gram matrix on the points $\{x_1, \ldots, x_n\}$. For this approach to be effective such kernel matrices must also take into account the distribution of unlabeled data, in order that the unlabeled data can aid in the classification task. Once these kernel matrices have been constructed, they can be deployed in standard kernel methods, for example support vector machines.

In this paper we motivate the construction of semi-supervised kernel matrices from a graph theoretic perspective. A graph is constructed where the nodes are the data instances $\{1, \ldots, n\}$ and an edge connects nodes $i$ and $j$ if a "local similarity" measure between $x_i$ and $x_j$ suggests they may have the same label. For example, the local similarity measure can be the Euclidean distance between feature vectors if $x \in \mathbb{R}^m$, and each node can connect to its $k$ nearest neighbors with weight value equal to 1. The intuition underlying the graph is that even if two nodes are not directly connected, they should be considered similar as long as there are many paths between them. Several semi-supervised learning algorithms have been proposed under the general graph theoretic theme, based on techniques such as random walks [8], diffusion kernels [5], and Gaussian fields [9]. Many of these methods can be unified into the regularization framework proposed by [7], which forms the basis of this paper.

The graph can be represented by an $n \times n$ weight matrix $W = [w_{ij}]$ where $w_{ij}$ is the edge weight between nodes $i$ and $j$, with $w_{ij} = 0$ if there is no edge. We require the entries of $W$ to be non-negative, and assume that it forms a symmetric matrix; it is not necessary for $W$ itself to be positive semi-definite. In semi-supervised learning $W$ is an essential quantity; we assume it is provided by domain experts, and hence do not study its construction. Let $D$ be a diagonal matrix where $d_{ii} = \sum_j w_{ij}$ is the degree of node $i$. This allows us to define the *combinatorial graph Laplacian* as $L = D - W$ (the normalized Laplacian $\tilde{L} = D^{-1/2} L D^{-1/2}$ can be used as well). We denote $L$'s eigensystem by $\{\lambda_i, \phi_i\}$, so that $L = \sum_{i=1}^n \lambda_i \phi_i \phi_i^\top$ where we assume the eigenvalues are sorted in non-decreasing order. The matrix $L$ has many interesting properties [3]; for instance, it is always positive

semi-definite, even if $W$ is not. Perhaps the most important property of the Laplacian related to semi-supervised learning is the following: a smaller eigenvalue $\lambda$ corresponds to a smoother eigenvector $\phi$ over the graph; that is, the value $\sum_{ij} w_{ij}(\phi(i) - \phi(j))^2$ is small. In a physical system the smoother eigenvectors correspond to the major vibration modes. Assuming the graph structure is correct, from a regularization perspective we want to encourage smooth functions, to reflect our belief that labels should vary slowly over the graph. Specifically, [2] and [7] suggest a general principle for creating a semi-supervised kernel $K$ from the graph Laplacian $L$: transform the eigenvalues $\lambda$ into $r(\lambda)$, where the *spectral transformation* $r$ is a non-negative and decreasing function[1]

$$K = \sum_{i=1}^{n} r(\lambda_i)\, \phi_i \phi_i^\top \tag{1}$$

Note that it may be that $r$ reverses the order of the eigenvalues, so that smooth $\phi_i$'s have *larger* eigenvalues in $K$. A "soft labeling" function $f = \sum c_i \phi_i$ in a kernel machine has a penalty term in the RKHS norm given by $\Omega(\|f\|_K^2) = \Omega(\sum c_i^2 / r(\lambda_i))$. Since $r$ is decreasing, a greater penality is incurred for those terms of $f$ corresponding to eigenfunctions that are less smooth. In previous work $r$ has often been chosen from a parametric family. For example, the diffusion kernel [5] corresponds to $r(\lambda) = \exp(-\frac{\sigma^2}{2}\lambda)$ and the Gaussian field kernel [10] corresponds to $r(\lambda) = \frac{1}{\lambda + \epsilon}$. Cross validation has been used to find the hyperparameters $\sigma$ or $\epsilon$ for these spectral transformations. Although the general principle of equation (1) is appealing, it does not address question of which parametric family to use for $r$. Moreover, the number of degrees of freedom (or the number of hyperparameters) may not suit the task at hand, resulting in overly constrained kernels. The contribution of the current paper is to address these limitations using a convex optimization approach by imposing an ordering constraint on $r$ but otherwise not assuming any parametric form for the kernels.

## 3  Convex Optimization using QCQP

Let $K_i = \phi_i \phi_i^\top, i = 1 \cdots n$ be the outer product matrices of the eigenvectors. The semi-supervised kernel $K$ is a linear combination $K = \sum_{i=1}^{n} \mu_i K_i$, where $\mu_i \geq 0$. We formulate the problem of finding the spectral transformation as one that finds the interpolation coefficients $\{r(\lambda_i) = \mu_i\}$ by optimizing some convex objective function on $K$. To maintain the positive semi-definiteness constraint on $K$, one in general needs to invoke SDPs [1]. Semi-definite optimization can be described as the problem of optimizing a linear function of a symmetric matrix subject to linear equality constraints and the condition that the matrix be positive semi-definite. The well-known linear programming problem can be generalized to a semi-definite optimization by replacing the vector of variables with a symmetric matrix, and replacing the non-negativity constraints with a positive semi-definite constraints. This generalization inherits several properties: it is convex, has a rich duality theory and allows theoretically efficient solution algorithms based on iterating interior point methods to either follow a central path or decrease a potential function. However, a limitation of SDPs is their computational complexity [1], which has restricted their application to small scale problems [6]. However, an important special case of SDPs are *quadratically constrained quadratic programs* (QCQP) which are computationally more efficient. Here both the objective function and the constraints are quadratic as illustrated below,

$$\text{minimize} \quad \frac{1}{2}x^\top P_0 x + q_0^\top x + r_0 \tag{2}$$

$$\text{subject to} \quad \frac{1}{2}x^\top P_i x + q_i^\top x + r_i \leq 0 \quad i = 1 \cdots m \tag{3}$$

$$Ax = b \tag{4}$$

where $P_i \in \mathcal{S}_+^n$, $i = 1, \ldots, m$, where $\mathcal{S}_+^n$ defines the set of square symmetric positive semi-definite matrices. In a QCQP, we minimize a convex quadratic function over a feasible region that is the intersection of ellipsoids. The number of iterations required to reach the solution is comparable to the number required for linear programs, making the approach feasible for large datasets. However, as observed in [1], not all SDPs can be relaxed to QCQPs. For the semi-supervised kernel learning task presented here solving an SDP would be computationally infeasible.

Recent work [4, 6] has proposed *kernel target alignment* that can be used not only to assess the relationship between the feature spaces generated by two different kernels, but also to assess the similarity between spaces induced by a kernel and that induced by the labels themselves. Desirable properties of the alignment measure can be found in [4]. The crucial aspect of alignnement for our purposes is that its optimization can be formulated as a QCQP. The objective function is the empirical kernel alignment score:

$$\hat{A}(K_{tr}, T) = \frac{\langle K_{tr}, T \rangle_F}{\sqrt{\langle K_{tr}, K_{tr} \rangle_F \langle T, T \rangle_F}} \tag{5}$$

where $K_{tr}$ is the kernel matrix restricted to the training points, $\langle M, N \rangle_F$ denotes the Frobenius product between two square matrices $\langle M, N \rangle_F = \sum_{ij} m_{ij} n_{ij} = Tr(MN^\top)$, and $T$ is the target matrix on training data, with entry $T_{ij}$ set to $+1$ if $y_i = y_j$ and $-1$ otherwise. Note for binary $\{+1, -1\}$ training labels $\mathbf{y}$ this is simply the rank one matrix $T = \mathbf{y}\mathbf{y}^\top$. $K$ is guaranteed to be positive semi-definite by constraining $\mu_i \geq 0$. Previous work using kernel alignment did not take into account that the $K_i$'s were derived from the graph Laplacian with the goal of semi-supervised learning. As such, the $\mu_i$'s can take arbitrary values and there is no preference to penalize components that do not vary smoothly over the graph. This can be rectified by requiring smoother eigenvectors to receive larger coefficients, as shown in the next section.

## 4 Semi-Supervised Kernels with Order Constraints

As stated above, we would like to maintain a decreasing order on the spectral transformation $\mu_i = r(\lambda_i)$ to encourage smooth functions over the graph. This motivates the set of *order constraints*

$$\mu_i \geq \mu_{i+1}, \quad i = 1 \cdots n - 1 \tag{6}$$

And we can specify the desired semi-supervised kernel as follows.

**Definition 1** *An order constrained semi-supervised kernel $K$ is the solution to the following convex optimization problem:*

$$\max_K \qquad \hat{A}(K_{tr}, T) \tag{7}$$
$$\textit{subject to} \qquad K = \sum_{i=1}^n \mu_i K_i \tag{8}$$
$$\mu_i \geq 0 \tag{9}$$
$$trace(K) = 1 \tag{10}$$
$$\mu_i \geq \mu_{i+1}, \quad i = 1 \cdots n - 1 \tag{11}$$

*where $T$ is the training target matrix, $K_i = \phi_i \phi_i^\top$ and $\phi_i$'s are the eigenvectors of the graph Laplacian.*

The formulation is an extension to [6] with order constraints, and with special components $K_i$'s from the graph Laplacian. Since $\mu_i \geq 0$ and $K_i$'s are outer products, $K$ will automatically be positive semi-definite and hence a valid kernel matrix. The trace constraint is needed to fix the scale invariance of kernel alignment. It is important to notice the order constraints are convex, and as such the whole problem is convex. Let $vec(A)$ be the column

vectorization of a matrix $A$. Defining $M = \begin{bmatrix} vec(K_{1,tr}) \cdots vec(K_{m,tr}) \end{bmatrix}$, it is not hard to show that the problem can then be expressed as

$$\max_\mu \qquad vec(T)^\top M \mu \qquad (12)$$

$$\text{subject to} \qquad ||M\mu|| \leq 1 \qquad (13)$$

$$\mu_i \geq 0 \qquad (14)$$

$$\mu_i \geq \mu_{i+1}, \qquad i = 1 \cdots n - 1 \qquad (15)$$

The objective function is linear in $\mu$, and there is a simple cone constraint, making it a quadratically constrained quadratic program (QCQP).

An improvement of the above order constrained semi-supervised kernel can be obtained by studying the Laplacian eigenvectors with zero eigenvalues. For a graph Laplacian there will be $k$ zero eigenvalues if the graph has $k$ connected subgraphs. The $k$ eigenvectors are piecewise constant over individual subgraphs, and zero elsewhere. This is desirable when $k > 1$, with the hope that subgraphs correspond to different classes. However if $k = 1$, the graph is connected. The first eigenvector $\phi_1$ is a constant vector. The corresponding $K_1$ is a constant matrix, and acts as a bias term. In this situation we do not want to impose the order constraint $\mu_1 \geq \mu_2$ on the constant bias term. Instead we let $\mu_1$ vary freely during optimization.

**Definition 2** *An improved order constrained semi-supervised kernel $K$ is the solution to the same problem in Definition 1, but the order constraints (11) apply only to non-constant eigenvectors:*

$$\mu_i \geq \mu_{i+1}, \qquad i = 1 \cdots n - 1, \text{ and } \phi_i \text{ not constant} \qquad (16)$$

In practice we do not need all $n$ eigenvectors of the graph Laplacian, or equivalently all $n$ $K_i$'s. The first $m < n$ eigenvectors with the smallest eigenvalues work well empirically. Also note we could have used the fact that $K_i$'s are from orthogonal eigenvectors $\phi_i$ to further simplify the expression. However we neglect this observation, making it easier to incorporate other kernel components if necessary.

It is illustrative to compare and contrast the order constrained semi-supervised kernels to other semi-supervised kernels with different spectral transformation. We call the original kernel alignment solution in [6] a *maximal-alignment* kernel. It is the solution to Definition 1 without the order constraints (11). Because it does not have the additional constraints, it maximizes kernel alignment among all spectral transformation. The hyperparameters $\sigma$ and $\epsilon$ of the Diffusion kernel and Gaussian fields kernel (described earlier) can be learned by maximizing the alignment score also, although the optimization problem is not necessarily convex. These kernels use different information from the original Laplacian eigenvalues $\lambda_i$. The maximal-alignment kernels ignore $\lambda_i$ altogether. The order constrained semi-supervised kernels only use the *order* of $\lambda_i$ and ignore their actual values. The diffusion and Gaussian field kernels use the actual values. In terms of the degree of freedom in choosing the spectral transformation $\mu_i$'s, the maximal-alignment kernels are completely free. The diffusion and Gaussian field kernels are restrictive since they have an implicit parametric form and only one free parameter. The order constrained semi-supervised kernels incorporates desirable features from both approaches.

## 5   Experimental Results

We evaluate the order constrained kernels on seven datasets. **baseball-hockey** (1993 instances / 2 classes), **pc-mac** (1943/2) and **religion-atheism** (1427/2) are document categorization tasks taken from the 20-newsgroups dataset. The distance measure is the standard cosine similarity between tf.idf vectors. **one-two** (2200/2), **odd-even** (4000/2) and **ten digits** (4000/10) are handwritten digits recognition tasks. **one-two** is digits "1" vs. "2"; **odd-even** is the artificial task of classifying odd "1, 3, 5, 7, 9" vs. even "0, 2, 4, 6, 8" digits,

such that each class has several well defined internal clusters; **ten digits** is 10-way classification. **isolet** (7797/26) is isolated spoken English alphabet recognition from the UCI repository. For these datasets we use Euclidean distance on raw features. We use 10NN unweighted graphs on all datasets except isolet which is 100NN. For all datasets, we use the smallest $m = 200$ eigenvalue and eigenvector pairs from the graph Laplacian. These values are set arbitrarily without optimizing and do not create a unfair advantage to the proposed kernels. For each dataset we test on five different labeled set sizes. For a given labeled set size, we perform 30 random trials in which a labeled set is randomly sampled from the whole dataset. All classes must be present in the labeled set. The rest is used as unlabeled (test) set in that trial. We compare 5 semi-supervised kernels (improved order constrained kernel, order constrained kernel, Gaussian field kernel, diffusion kernel[2] and maximal-alignment kernel), and 3 standard supervised kernels (RBF (bandwidth learned using 5-fold cross validation),linear and quadratic). We compute the spectral transformation for order constrained kernels and maximal-alignment kernels by solving the QCQP using standard solvers (SeDuMi/YALMIP). To compute accuracy we use a standard SVM. We choose the the bound on slack variables $C$ with cross validation for all tasks and kernels. For multiclass classification we perform one-against-all and pick the class with the largest margin.

The results[3] are shown in Table 1, which has two rows for each cell: The upper row is the average *test set accuracy* with one standard deviation; The lower row is the average *training set kernel alignment*, and in parenthesis the average *run time in seconds* for SeDuMi/YALMIP on a 3GHz Linux computer. Each number is averaged over 30 random trials. To assess the statistical significance of the results, we perform paired $t$-test on test accuracy. We highlight the best accuracy in each row, and those that can not be determined as different from the best, with paired $t$-test at significance level 0.05. The semi-supervised kernels tend to outperform standard supervised kernels. The improved order constrained kernels are consistently among the best. Figure 1 shows the spectral transformation $\mu_i$ of the semi-supervised kernels for different tasks. These are for the 30 trials with the largest labeled set size in each task. The $x$-axis is in increasing order of $\lambda_i$ (the original eigenvalues of the Laplacian). The mean (thick lines) and $\pm 1$ standard deviation (dotted lines) of only the top 50 $\mu_i$'s are plotted for clarity. The $\mu_i$ values are scaled vertically for easy comparison among kernels. As expected the maximal-alignment kernels' spectral transformation is zigzagged, diffusion and Gaussian field's are very smooth, while order constrained kernels' are in between. The order constrained kernels (green) have large $\mu_1$ because of the order constraint. This seems to be disadvantageous — the spectral transformation tries to balance it out by increasing the value of other $\mu_i$'s so that the constant $K_1$'s relative influence is smaller. On the other hand the improved order constrained kernels (black) allow $\mu_1$ to be small. As a result the rest $\mu_i$'s decay fast, which is desirable.

## 6  Conclusions

We have proposed and evaluated a novel approach for semi-supervised kernel construction using convex optimization. The method incorporates order constraints, and the resulting convex optimization problem can be solved efficiently using a QCQP. In this work the *base* kernels were derived from the graph Laplacian, and no parametric form for the spectral transformation was imposed, making the approach more general than previous approaches. Experiments show that the method is both computationally feasible and results in improvements to classification performance when used with support vector machines.

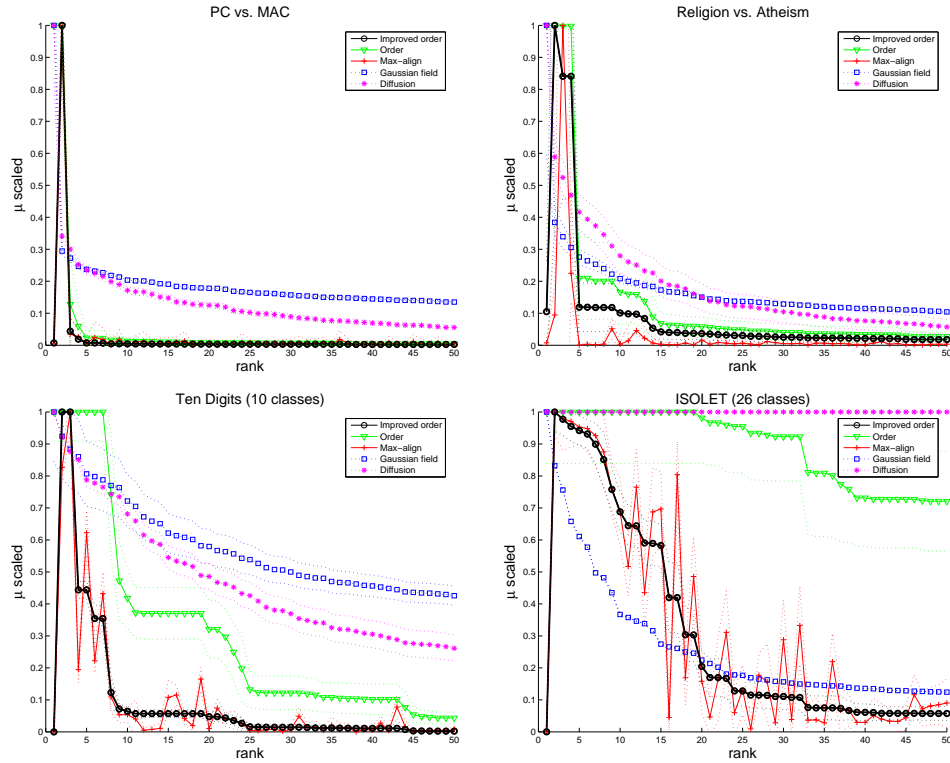

Figure 1: Comparison of spectral transformation for the 5 semi-supervised kernels.

## Footnotes

[1] We use a slightly different notation where $r$ is the inverse of that in [7].

[2]The hyperparameters $\sigma^2$ and $\epsilon$ are learned with the `fminbnd()` function in Matlab to maximize kernel alignment.

[3]Results on baseball-hockey and odd-even are similar and omitted for space. Full results can be found at `http://www.cs.cmu.edu/~zhuxj/pub/ocssk.pdf`

## References

[1] S. Boyd and L. Vandenberge. *Convex Optimization*. Cambridge University Press, Cambridge UK, 2004.

[2] O. Chapelle, J. Weston, and B. Schölkopf. Cluster kernels for semi-supervised learning. In *Advances in Neural Information Processing Systems, 15*, volume 15, 2002.

[3] F. R. K. Chung. *Spectral graph theory, Regional Conference Series in Mathematics, No. 92*. American Mathematical Society, 1997.

[4] N. Cristianini, J. Shawe-Taylor, A. Elisseeff, and J. Kandola. On kernel-target alignment. In *Advances in NIPS*, 2001.

[5] R. I. Kondor and J. Lafferty. Diffusion kernels on graphs and other discrete input spaces. In *Proc. 19th International Conf. on Machine Learning*, 2002.

[6] G. Lanckriet, N. Cristianini, P. Bartlett, L. E. Ghaoui, and M. Jordan. Learning the kernel matrix with semidefinite programming. *Journal of Machine Learning Research*, 5:27–72, 2004.

[7] A. Smola and R. Kondor. Kernels and regularization on graphs. In *Conference on Learning Theory, COLT/KW*, 2003.

[8] M. Szummer and T. Jaakkola. Partially labeled classification with Markov random walks. In *Advances in Neural Information Processing Systems, 14*, volume 14, 2001.

[9] X. Zhu, Z. Ghahramani, and J. Lafferty. Semi-supervised learning using Gaussian fields and harmonic functions. In *ICML-03, 20th International Conference on Machine Learning*, 2003.

[10] X. Zhu, J. Lafferty, and Z. Ghahramani. Semi-supervised learning: From Gaussian fields to Gaussian processes. Technical Report CMU-CS-03-175, Carnegie Mellon University, 2003.

| | semi-supervised kernels | | | | | standard kernels | | |
|---|---|---|---|---|---|---|---|---|
| Training set size | Improved Order | Order | Gaussian Field | Diffusion | Max-align | RBF | Linear | Quadratic |
| **pc-mac** | | | | | | | | |
| 10 | **87.0** ± 5.0 | **84.9** ± 7.2 | 56.4 ± 6.2 | 57.8 ±11.5 | 71.1 ± 9.7 | 51.6 ± 3.4 | 63.0 ± 5.1 | 62.3 ± 4.2 |
| | 0.71 ( 1) | 0.57 ( 1) | 0.32 | 0.35 | 0.90 ( 1) | 0.11 | 0.30 | 0.25 |
| 30 | **90.3** ± 1.3 | 89.6 ± 2.3 | 76.4 ± 6.1 | 79.6 ±11.2 | 85.4 ± 3.9 | 62.6 ± 9.6 | 71.8 ± 5.5 | 71.2 ± 5.3 |
| | 0.68 ( 8) | 0.49 ( 8) | 0.19 | 0.23 | 0.74 ( 6) | 0.03 | 0.18 | 0.13 |
| 50 | **91.3** ± 0.9 | 90.5 ± 1.7 | 81.1 ± 4.6 | 87.5 ± 2.8 | 88.4 ± 2.1 | 67.8 ± 9.0 | 77.6 ± 4.8 | 75.7 ± 5.4 |
| | 0.64 (31) | 0.46 (31) | 0.16 | 0.20 | 0.68 (25) | 0.02 | 0.14 | 0.10 |
| 70 | **91.5** ± 0.6 | 90.8 ± 1.3 | 84.6 ± 2.1 | 90.5 ± 1.2 | 89.6 ± 1.6 | 74.7 ± 7.4 | 80.2 ± 4.6 | 74.3 ± 8.7 |
| | 0.63 (70) | 0.46 (56) | 0.14 | 0.19 | 0.66 (59) | 0.01 | 0.12 | 0.08 |
| 90 | **91.5** ± 0.6 | **91.3** ± 1.3 | 86.3 ± 2.3 | **91.3** ± 1.1 | 90.3 ± 1.0 | 79.0 ± 6.4 | 82.5 ± 4.2 | 79.1 ± 7.3 |
| | 0.63 (108) | 0.45 (98) | 0.13 | 0.18 | 0.65 (84) | 0.01 | 0.11 | 0.08 |
| **religion-atheism** | | | | | | | | |
| 10 | **72.8** ±11.2 | 70.9 ±10.9 | 55.2 ± 5.8 | 60.9 ±10.7 | 60.7 ± 7.5 | 55.8 ± 5.8 | 60.1 ± 7.0 | 61.2 ± 4.8 |
| | 0.50 ( 1) | 0.42 ( 1) | 0.31 | 0.31 | 0.85 ( 1) | 0.13 | 0.30 | 0.26 |
| 30 | **84.2** ± 2.4 | 83.0 ± 2.9 | 71.2 ± 6.3 | 80.3 ± 5.1 | 74.4 ± 5.4 | 63.4 ± 6.5 | 63.7 ± 8.3 | 70.1 ± 6.3 |
| | 0.38 ( 8) | 0.31 ( 6) | 0.20 | 0.22 | 0.60 ( 7) | 0.05 | 0.18 | 0.15 |
| 50 | **84.5** ± 2.3 | 83.5 ± 2.5 | 80.4 ± 4.1 | 83.5 ± 2.7 | 77.4 ± 6.1 | 69.3 ± 6.5 | 69.4 ± 7.0 | 70.7 ± 8.5 |
| | 0.31 (28) | 0.26 (23) | 0.17 | 0.20 | 0.48 (27) | 0.04 | 0.15 | 0.11 |
| 70 | **85.7** ± 1.4 | 85.3 ± 1.6 | 83.0 ± 2.9 | **85.4** ± 1.8 | 82.3 ± 3.0 | 73.1 ± 5.8 | 75.7 ± 6.0 | 71.0 ±10.0 |
| | 0.29 (55) | 0.25 (42) | 0.16 | 0.19 | 0.43 (51) | 0.03 | 0.13 | 0.10 |
| 90 | **86.6** ± 1.3 | **86.4** ± 1.5 | 84.5 ± 2.1 | **86.2** ± 1.6 | 82.8 ± 2.6 | 77.7 ± 5.1 | 74.6 ± 7.6 | 70.0 ±11.5 |
| | 0.27 (86) | 0.24 (92) | 0.15 | 0.18 | 0.40 (85) | 0.02 | 0.12 | 0.09 |
| **one-two** | | | | | | | | |
| 10 | **96.2** ± 2.7 | 90.6 ±14.0 | 58.2 ±17.6 | 59.4 ±18.9 | 85.4 ±11.5 | 78.7 ±14.3 | 85.1 ± 5.7 | 85.7 ± 4.8 |
| | 0.87 ( 2) | 0.66 ( 1) | 0.43 | 0.53 | 0.95 ( 1) | 0.38 | 0.26 | 0.30 |
| 20 | **96.4** ± 2.8 | 93.9 ± 8.7 | 87.0 ±16.0 | 83.2 ±19.8 | 94.5 ± 1.6 | 90.4 ± 4.6 | 86.0 ± 9.4 | 90.9 ± 3.7 |
| | 0.87 ( 3) | 0.64 ( 4) | 0.38 | 0.50 | 0.90 ( 4) | 0.33 | 0.22 | 0.25 |
| 30 | **98.2** ± 2.1 | 97.2 ± 2.5 | **98.1** ± 2.2 | **98.1** ± 2.7 | 96.4 ± 2.1 | 93.6 ± 3.1 | 89.6 ± 5.9 | 92.9 ± 2.8 |
| | 0.84 ( 8) | 0.61 ( 7) | 0.35 | 0.47 | 0.86 ( 6) | 0.30 | 0.17 | 0.24 |
| 40 | 98.3 ± 1.9 | 96.5 ± 2.4 | 98.9 ± 1.8 | **99.1** ± 1.4 | 96.3 ± 2.3 | 94.0 ± 2.7 | 91.6 ± 6.3 | 94.9 ± 2.0 |
| | 0.84 (13) | 0.61 (15) | 0.36 | 0.48 | 0.86 (11) | 0.29 | 0.18 | 0.21 |
| 50 | 98.4 ± 1.9 | 95.6 ± 9.0 | 99.4 ± 0.5 | **99.6** ± 0.3 | 96.6 ± 2.3 | 96.1 ± 2.4 | 93.0 ± 3.6 | 95.8 ± 2.3 |
| | 0.83 (31) | 0.60 (37) | 0.35 | 0.46 | 0.84 (25) | 0.28 | 0.17 | 0.20 |
| **Ten digits (10 classes)** | | | | | | | | |
| 50 | **76.6** ± 4.3 | 71.5 ± 5.0 | 41.4 ± 6.8 | 49.8 ± 6.3 | 70.3 ± 5.2 | 57.0 ± 4.0 | 50.2 ± 9.0 | 66.3 ± 3.7 |
| | 0.47 (26) | 0.21 (26) | 0.15 | 0.16 | 0.51 (25) | -0.62 | -0.50 | -0.25 |
| 100 | **84.8** ± 2.6 | 83.4 ± 2.6 | 63.7 ± 3.5 | 72.5 ± 3.3 | 80.7 ± 2.6 | 69.4 ± 1.9 | 56.0 ± 7.8 | 77.2 ± 2.3 |
| | 0.47 (124) | 0.17 (98) | 0.12 | 0.13 | 0.49 (100) | -0.64 | -0.52 | -0.29 |
| 150 | **86.5** ± 1.7 | **86.4** ± 1.3 | 75.1 ± 3.0 | 80.4 ± 2.1 | 84.5 ± 1.9 | 75.2 ± 1.4 | 56.2 ± 7.2 | 81.4 ± 2.2 |
| | 0.48 (310) | 0.18 (255) | 0.11 | 0.13 | 0.50 (244) | -0.66 | -0.53 | -0.31 |
| 200 | **88.1** ± 1.3 | **88.0** ± 1.3 | 80.4 ± 2.5 | 84.4 ± 1.6 | 86.0 ± 1.5 | 78.3 ± 1.3 | 60.8 ± 7.3 | 84.3 ± 1.7 |
| | 0.47 (708) | 0.16 (477) | 0.10 | 0.11 | 0.49 (523) | -0.65 | -0.54 | -0.33 |
| 250 | **89.1** ± 1.1 | **89.3** ± 1.0 | 84.6 ± 1.4 | 87.2 ± 1.3 | 87.2 ± 1.3 | 80.4 ± 1.4 | 61.3 ± 7.6 | 85.7 ± 1.3 |
| | 0.47 (942) | 0.16 (873) | 0.10 | 0.11 | 0.49 (706) | -0.65 | -0.54 | -0.33 |
| **isolet (26 classes)** | | | | | | | | |
| 50 | **56.0** ± 3.5 | 42.0 ± 5.2 | 41.2 ± 2.9 | 29.0 ± 2.7 | 50.1 ± 3.7 | 28.7 ± 2.0 | 30.0 ± 2.7 | 23.7 ± 2.4 |
| | 0.27 (26) | 0.13 (25) | 0.03 | 0.11 | 0.31 (24) | -0.89 | -0.80 | -0.65 |
| 100 | **64.6** ± 2.1 | 59.0 ± 3.6 | 58.5 ± 2.9 | 47.4 ± 2.7 | 63.2 ± 1.9 | 46.3 ± 2.4 | 46.6 ± 2.7 | 42.0 ± 2.9 |
| | 0.26 (105) | 0.10 (127) | -0.02 | 0.08 | 0.29 (102) | -0.90 | -0.82 | -0.69 |
| 150 | 67.6 ± 2.6 | 65.2 ± 3.0 | 65.4 ± 2.6 | 57.2 ± 2.7 | **67.9** ± 2.5 | 57.6 ± 1.5 | 57.3 ± 1.8 | 53.8 ± 2.2 |
| | 0.26 (249) | 0.09 (280) | -0.05 | 0.07 | 0.27 (221) | -0.90 | -0.83 | -0.70 |
| 200 | 71.0 ± 1.8 | 70.9 ± 2.3 | 70.6 ± 1.9 | 64.8 ± 2.1 | **72.3** ± 1.7 | 63.9 ± 1.6 | 64.2 ± 2.0 | 60.5 ± 1.6 |
| | 0.26 (441) | 0.08 (570) | -0.07 | 0.06 | 0.27 (423) | -0.91 | -0.83 | -0.72 |
| 250 | 71.8 ± 2.3 | 73.6 ± 1.5 | 73.7 ± 1.2 | 69.8 ± 1.5 | **74.2** ± 1.5 | 68.8 ± 1.5 | 69.5 ± 1.7 | 66.2 ± 1.4 |
| | 0.26 (709) | 0.08 (836) | -0.07 | 0.06 | 0.27 (665) | -0.91 | -0.84 | -0.72 |

Table 1: Accuracy, alignment scores, and run times on the datasets. The table compares 8 kernels. Each cell has two rows: The upper row is test set accuracy with standard error; the lower row is training set alignment (SeDuMi/YALMIP run time in seconds is given in parentheses). All numbers are averaged over 30 random trials. Accuracies in boldface are the best as determined by a paired $t$-test at the 0.05 significance level.